# Humans Learn Using Manifolds, Reluctantly

**Bryan R. Gibson, Xiaojin Zhu, Timothy T. Rogers**[*]**, Charles W. Kalish**[†]**, Joseph Harrison**[*]
Department of Computer Sciences, [*]Psychology, and [†]Educational Psychology
University of Wisconsin-Madison, Madison, WI 53706 USA
{bgibson, jerryzhu}@cs.wisc.edu
{ttrogers, cwkalish, jcharrison}@wisc.edu

## Abstract

When the distribution of unlabeled data in feature space lies along a manifold, the information it provides may be used by a learner to assist classification in a semi-supervised setting. While manifold learning is well-known in machine learning, the use of manifolds in human learning is largely unstudied. We perform a set of experiments which test a human's ability to use a manifold in a semi-supervised learning task, under varying conditions. We show that humans may be encouraged into using the manifold, overcoming the strong preference for a simple, axis-parallel linear boundary.

## 1 Introduction

Consider a classification task where a learner is given training items $x_1, \ldots, x_l \in \mathbb{R}^d$, represented by $d$-dimensional feature vectors. The learner is also given the corresponding class labels $y_1, \ldots, y_l \in \mathcal{Y}$. In this paper, we focus on binary labels $\mathcal{Y} \in \{-1, 1\}$. In addition, the learner is given some unlabeled items $x_{l+1}, \ldots, x_{l+u} \in \mathbb{R}^d$ without the corresponding labels. Importantly, the labeled and unlabeled items $x_1 \ldots x_{l+u}$ are distributed in a peculiar way in the feature space: they lie on smooth, lower dimension *manifolds*, such as those schematically shown in Figure 1(a). The question is: given this knowledge of labeled and unlabeled data, how will the learner classify $x_{l+1}, \ldots, x_{l+u}$? Will the learner ignore the distribution information of the unlabeled data, and simply use the labeled data to form a decision boundary as in Figure 1(b)? Or will the learner propagate labels along the nonlinear manifolds as in Figure 1(c)?

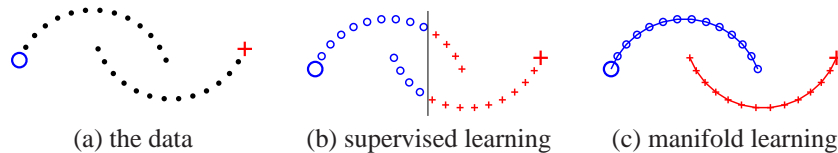

|  (a) the data  |  (b) supervised learning  |  (c) manifold learning  |

Figure 1: On a dataset with manifold structure, supervised learning and manifold learning make dramatically different predictions. Large symbols represent labeled items, dots unlabeled items.

When the learner is a machine learning algorithm, this question has been addressed by semi-supervised learning [2, 11]. The designer of the algorithm can *choose* to make the manifold assumption, also known as graph-based semi-supervised learning, which states that the labels vary slowly along the manifolds or the discrete graph formed by connecting nearby items. Consequently, the learning algorithm will predict Figure 1(c). The mathematics of manifold learning is well-understood [1, 6, 9, 10]. Alternatively, the designer can choose to ignore the unlabeled data and perform supervised learning, which results in Figure 1(b).

When the learner is a human being, however, the answer is not so clear. Consider that the human learner does not directly see how the items are distributed in the feature space (such as Figure 1(a)), but only a set of items (such as those in Figure 2(a)). The underlying manifold structure of the data may not be immediately obvious. Thus there are many possibilities for how the human learner will behave: 1) They may completely ignore the manifold structure and perform supervised learning; 2) They may discover the manifold under some learning conditions and not others; or 3) They may always learn using the manifold.

For readers not familiar with manifold learning, the setting might seem artificial. But in fact, many natural stimuli we encounter in everyday life are distributed on manifolds. An important example is face recognition, where different poses (viewing angles) of the same face produce different 2D images. These images can be quite different, as in the frontal and profile views of a person. However, if we continuously change the viewing angle, these 2D images will form a one-dimensional manifold in a very high dimensional image space. This example illustrates the importance of a manifold to facilitate learning: if we can form and maintain such a face manifold, then with a single label (e.g., the name) on one of the face images, we can recognize all other poses of that person by propagating the label along the manifold. The same is true for visual object recognition in general. Other more abstract stimuli form manifolds, or the discrete analogue, graphs. For example, text documents in a corpus occupy a potentially nonlinear manifold in the otherwise very high dimensional space used to represent them, such as the "bag of words" representation.

There exists little empirical evidence addressing the question of whether human beings can learn using manifolds when classifying objects, and the few studies we are aware of come to opposing conclusions. For instance, Wallis and Bülthoff created artificial image sequences where a frontal face is morphed into the profile face of a different person. When participants were shown such sequences during training, their ability to match frontal and profile faces during testing was impaired [8]. This might be evidence that people depend on manifold structure stemming from temporal and spatial proximity to perform face recognition. On the other hand, Vandist *et al.* conducted a categorization experiment where the true decision boundary is at 45 degrees in a 2D stimulus space (i.e., an information integration task). They showed that when the two classes are elongated Gaussian, which are parallel to, and on opposite sides of, the decision boundary, unlabeled data does not help learning [7]. If we view these two elongated Gaussian as linear manifolds, this result suggests that people do not generally learn using manifolds.

This study seeks to understand under what conditions, if any, people are capable of manifold learning in a semi-supervised setting. The study has important implications for cognitive psychology: first, if people are capable of learning manifolds, this suggests that manifold-learning models that have been developed in machine learning can provide hypotheses about how people categorize objects in natural domains like face recognition, where manifolds appear to capture the true structure of the domain. Second, if there are reliable methods for encouraging manifold learning in people, these methods can be employed to aid learning in other domains that are structured along manifolds. For machine learning, our study will help in the design of algorithms which can decide when to invoke the manifold learning assumption.

## 2   Human Manifold Learning Experiments

We designed and conducted a set of experiments to study manifold learning in humans, with the following design considerations. First, the task was a "batch learning" paradigm in which participants viewed all labeled and unlabeled items at once (in contrast to "online" or sequential learning paradigm where items appear one at a time). Batch learning allows us to compare human behavior against well-established machine learning models that typically operate in batch mode. Second, we avoided using faces or familiar 3D objects as stimuli, despite their natural manifold structures as discussed above, because we wished to avoid any bias resulting from strong prior real-world knowledge. Instead, we used unfamiliar stimuli, from which we could add or remove a manifold structure easily. This design should allow our experiments to shed light on people's intrinsic ability to learn using a manifold.

**Participants and Materials.** In the first set of experiments, 139 university undergraduates participated for partial course credit. A computer interface was created to represent a table with three bins, as shown in Figure 2(a). Unlabeled cards were initially placed in a central white bin, with bins to

either side colored red and blue to indicate the two classes $y \in \{-1, 1\}$. Each stimulus is a card. Participants sorted cards by clicking and dragging with a mouse. When a card was clicked, other similar cards could be "highlighted" in gray (depending on condition). Labeled cards were pinned down in their respective red or blue bins and could not be moved, indicated by a "pin" in the corner of the card. The layout of the cards was such that all cards remained visible at all times. Unlabeled cards could be re-categorized at any time by dragging from any bin to any other bin. Upon sorting all cards, participants would click a button to indicating completion.

Two sets of stimuli were created. The first, used solely to acquaint the participants with the interface, consisted of a set of 20 cards with animal line drawings on a white background. The images were chosen to approximate a linear continuum between fish and mammal, with shark, dolphin, and whale at the center. The second set of stimuli used for the actual experiment was composed of 82 "crosshair" cards, each with a pair of perpendicular, axis-parallel lines, all of equal length, crossing on a white background. Four examples are shown in Figure 2(b). Each card therefore can be encoded as $x \in [0, 1]^2$, whose two features representing the positions of the vertical and horizontal lines, respectively.

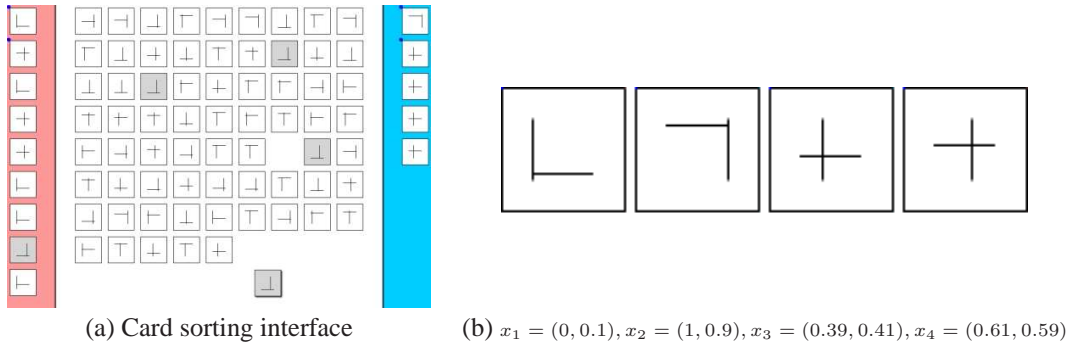

(a) Card sorting interface　　(b) $x_1 = (0, 0.1), x_2 = (1, 0.9), x_3 = (0.39, 0.41), x_4 = (0.61, 0.59)$

Figure 2: Experimental interface (with highlighting shown), and example crosshair stimuli.

**Procedure.** Each participant was given two tasks to complete.

Task 1 was a practice task to familiarize the participant with the interface. The participant was asked to sort the set of 20 animal cards into two categories, with the two ends of the continuum (a clown fish and a dachshund) labeled. Participants were told that when they clicked on a card, highlighting of similar cards might occur. In reality, highlighting was always shown for the two nearest-neighboring cards (on the defined continuum) of a clicked card. Importantly, we designed the dataset so that, near the middle of the continuum, cards from opposite biological classes would be highlighted together. For example, when a dolphin was clicked, both a shark and a whale would be highlighted. The intention was to indicate to the participant that highlighting is not always a clear give-away for class labels. At the end of task 1 their fish vs. mammal classification accuracy was presented. No time limit was enforced.

Task 2 asked the participant to sort a set of 82 crosshair cards into two categories. The set of cards, the number of labeled cards, and the highlighting of cards depended on condition. The participant was again told that some cards might be highlighted, whether the condition actually provided for highlighting or not. The participant was also told that cards that shared highlighting may not all have the same classification. Again, no time limit was enforced. After they completed this task, a follow up questionnaire was administered.

**Conditions.** Each of the 139 participants was randomly assigned to one of 6 conditions, shown in Figure 3, which varied according to three manipulations:

**The number of labeled items $l$ can be 2 or 4 ($2^l$ vs. $4^l$).** For conditions with two labeled items, the labeled items are always $(x_1, y_1 = -1), (x_2, y_2 = 1)$; with four labeled items, they are always $(x_1, y_1 = -1), (x_2, y_2 = 1), (x_3, y_3 = 1), (x_4, y_4 = -1)$. The features of $x_1 \ldots x_4$ are those given in Figure 2(b). We chose these four labeled points by maximizing the prediction differences made by seven machine learning models, as discussed in the next section.

**Unlabeled items are distributed on a uniform grid or manifolds (grid$^U$ vs. moons$^U$).** The items $x_5 \ldots x_{82}$ were either on a uniform grid in the 2D feature space, or along two "half-moons", which is a well-studied dataset in the semi-supervised learning community. No linear boundary can separate the two moons in feature space. $x_3$ and $x_4$, if unlabeled, are the same as in Figure 2(b).

**Highlighting similar items or not (the suffix h).** For the moons$^U$conditions, the neighboring cards of any clicked card may be highlighted. The neighborhood is defined as within a radius of $\epsilon = 0.07$ in the Euclidean feature space. This value was chosen as it includes at least two neighbors for each point in the moons$^U$dataset. To form the unweighted graph shown in Figure 3, an edge is placed between all neighboring points.

The rationale for comparing these different conditions will become apparent as we consider how different machine-learning models perform on these datasets.

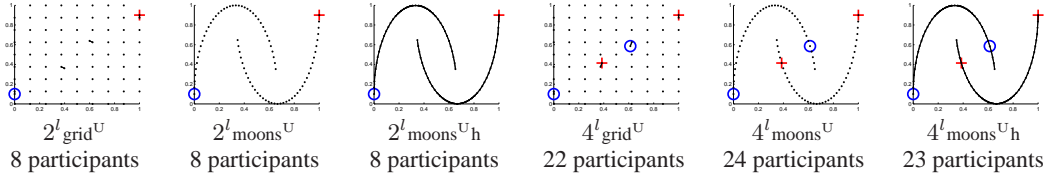

| $2^l$grid$^U$ | $2^l$moons$^U$ | $2^l$moons$^U$h | $4^l$grid$^U$ | $4^l$moons$^U$ | $4^l$moons$^U$h |
|---|---|---|---|---|---|
| 8 participants | 8 participants | 8 participants | 22 participants | 24 participants | 23 participants |

Figure 3: The six experimental conditions. Large symbols indicate labeled items, dots unlabeled items. Highlighting is represented as graph edges.

## 3    Model Predictions

We hypothesize that human participants consider a set of models ranging from simple to sophisticated, and that they will perform model selection based on the training data given to them. We start by considering seven typical machine learning models to motivate our choice, and present the models we actually use later on. The seven models are: **(graph)** Graph-based semi-supervised learning [1, 10], which propagates labels along the graph. It reverts to supervised learning when there is no graph (i.e., no highlighting). **(1NN,$\ell_2$)** 1-nearest-neighbor classifier with $\ell_2$ (Euclidean) distance. **(1NN,$\ell_1$)** 1-nearest-neighbor classifier with $\ell_1$ (Manhattan) distance. These two models are similar to exemplar models in psychology [3]. **(multi-v)** multiple vertical linear boundaries. **(multi-h)** multiple horizontal linear boundaries. **(single-v)** a single vertical linear boundary. **(single-h)** a single horizontal linear boundary. We plot the label predictions by these 7 models on four of the six conditions in Figure 4. Their predictions on $2^l$moons$^U$are identical to $2^l$moons$^U$h, and on $4^l$moons$^U$are identical to $4^l$moons$^U$h, except that "(graph)" is not available.

For conceptual simplicity and elegance, instead of using these disparate models we adopt a single model capable of making all these predictions. In particular, we use a Gaussian Process (GP) with different kernels (i.e., covariance functions) $k$ to simulate the seven models. For details on GPs see standard textbooks such as [4]. In particular, we find seven different kernels $k$ to match GP classification to each of the seven model predictions on all 6 conditions. This is somewhat unusual in that our GPs are not learned from data, but by matching other model predictions. Nonetheless, it is a valid procedure to create seven different GPs which will later be compared against human data.

For models (1NN,$\ell_2$), (multi-v), (multi-h), (single-v), and (single-h), we use diagonal RBF kernels $diag(\sigma_1^2, \sigma_2^2)$ and tune $\sigma_1, \sigma_2$ on a coarse parameter grid to minimize classification disagreement w.r.t. the corresponding model prediction on all 6 conditions. For model (1NN,$\ell_1$) we use a Laplace kernel and tune its bandwidth. For model (graph), we produce a graph kernel $\tilde{k}$ following the Reproducing Kernel Hilbert Space trick in [6]. That is, we extend a base RBF kernel $k$ with a graph component:

$$\tilde{k}(x, z) = k(x, z) - \mathbf{k}_x^\top (I + cLK)^{-1} cL\mathbf{k}_z \qquad (1)$$

where $x, z$ are two arbitrary items (not necessarily on the graph), $\mathbf{k}_x = (k(x, x_1), \ldots, k(x, x_{l+u}))^\top$ is the kernel vector between $x$ and all $l + u$ points $x_1 \ldots x_{l+u}$ in the graph, $K$ is the $(l + u) \times (l + u)$ Gram matrix with $K_{ij} = k(x_i, x_j)$, $L$ is the unnormalized graph Laplacian matrix derived from unweighted edges on the $\epsilon$NN graph defined earlier for highlighting, and $c$ is the parameter that we tune. We take the base RBF kernel $k$ to be the tuned kernel for model (1NN,$\ell_2$). It can be shown that

$\tilde{k}$ is a valid kernel formed by warping the base kernel $k$ along the graph, see [6] for technical details. We used the GP classification implementation with Expectation Propagation approximation [5].

In the end, our seven GPs were able to *exactly* match the predictions made by the seven models in Figure 4. We will use these GPs in the rest of the paper.

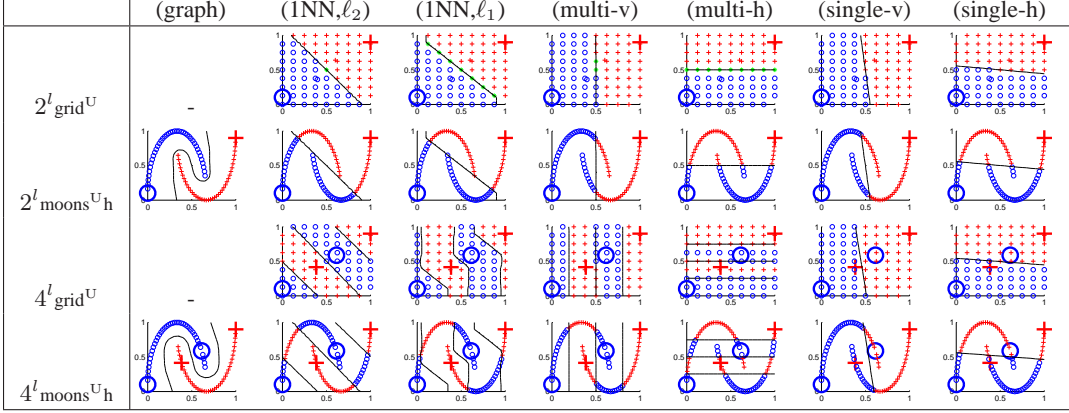

Figure 4: Predictions made by the seven models on 4 of the 6 conditions.

## 4   Behavioral Experiment Results

We now compare human categorization behaviors to model predictions. We first consider the aggregate behavior for all participants within each condition. One way to characterize this aggregate behavior is the "majority vote" of the participants on each item. That is, if more than half of the participants classified an item as $y = 1$, the majority vote classification for that item is $y = 1$, and so on. The first row in Figure 5 shows the majority vote for each condition. In these and all further plots, blue circles indicate $y = -1$, red pluses $y = 1$, and green stars ambiguous, meaning the classification into positive or negative is half-half. We also compute how well the seven GPs predict human majority votes. The accuracies of these GP models are shown in Table 1[1].

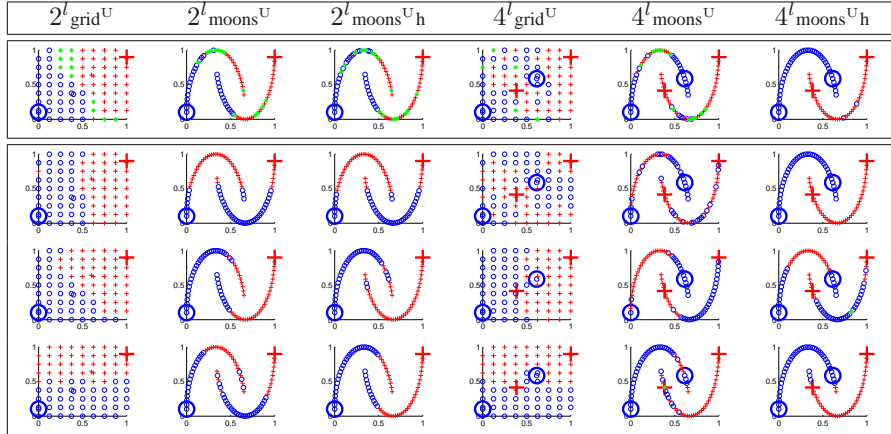

Figure 5: Human categorization results. (First row) the majority vote of participants within each condition. (Bottom three rows) a sample of responses from 18 different participants.

Of course, a majority vote only reveals average behavior. We have observed that there are wide participant variabilities. Participants appeared to find the tasks difficult, as their self-reported confidence scores were fairly low in all conditions. It was also noted that strategies for completing the

| | (graph) | (1NN,$\ell_2$) | (1NN,$\ell_1$) | (multi-v) | (multi-h) | (single-v) | (single-h) |
|---|---|---|---|---|---|---|---|
| $2^l$grid$^U$ | 0.81 | **0.94** | 0.84 | 0.86 | 0.58 | 0.85 | 0.61 |
| $2^l$moons$^U$ | 0.47 | **0.84** | 0.62 | 0.74 | 0.42 | 0.79 | 0.45 |
| $2^l$moons$^U$h | 0.50 | **0.78** | 0.56 | 0.76 | 0.36 | 0.76 | 0.39 |
| $4^l$grid$^U$ | 0.54 | 0.61 | **0.64** | **0.64** | 0.50 | 0.60 | 0.51 |
| $4^l$moons$^U$ | 0.64 | 0.62 | 0.60 | **0.69** | 0.47 | 0.38 | 0.45 |
| $4^l$moons$^U$h | **0.97** | 0.76 | 0.54 | 0.64 | 0.31 | 0.65 | 0.26 |
| $4^l$moons$^U$h$^R$ | **0.68** | 0.63 | 0.44 | 0.56 | 0.40 | 0.59 | 0.42 |

Table 1: GP model accuracy in predicting human majority vote for each condition.

task varied widely, with some participant simply categorizing cards in the order they appeared on the screen, while others took a much longer, studied approach. Most interestingly, different participants seem to use different models, as the individual participant plots in the bottom three rows of Figure 5 suggest. We would like to be able to make a claim about what model, from our set of models, each participant used for classification. In order to do this, we compute *per participant* accuracies of the seven models on that participant's classification. We then find the model $M$ with the highest accuracy for the participant, out of the seven models. If this highest accuracy is above 0.75, we declare that the participant is potentially using model $M$; otherwise no model is deemed a good fit and we say the participant is using some "other" model. We show the proportion of participants in each condition attributed to each of our seven models, plus "other", in Table 2.

| | (graph) | (1NN,$\ell_2$) | (1NN,$\ell_1$) | (multi-v) | (multi-h) | (single-v) | (single-h) | other |
|---|---|---|---|---|---|---|---|---|
| $2^l$grid$^U$ | 0.12 | 0.00 | 0.12 | **0.25** | **0.25** | 0.12 | 0.00 | 0.12 |
| $2^l$moons$^U$ | 0.00 | 0.12 | 0.00 | **0.25** | **0.25** | **0.25** | 0.00 | 0.12 |
| $2^l$moons$^U$h | 0.12 | 0.00 | 0.00 | **0.38** | 0.25 | 0.00 | 0.00 | 0.25 |
| $4^l$grid$^U$ | 0.00 | 0.05 | 0.09 | 0.00 | 0.00 | 0.18 | 0.09 | **0.59** |
| $4^l$moons$^U$ | 0.25 | 0.25 | 0.12 | 0.12 | 0.00 | 0.04 | 0.08 | **0.38** |
| $4^l$moons$^U$h | **0.39** | 0.09 | 0.09 | 0.04 | 0.04 | 0.00 | 0.13 | 0.22 |
| $4^l$moons$^U$h$^R$ | 0.13 | 0.03 | 0.07 | 0 | 0 | 0.07 | 0.03 | **0.67** |

Table 2: Percentage of participants potentially using each model

Based on Figure 5, Table 1, and Table 2, we make some observations:

**1.** When there are only two labeled points, the unlabeled distribution does not encourage humans to perform manifold learning (comparing $2^l$grid$^U$ vs. $2^l$moons$^U$). That is, they do not follow the possible implicit graph structure ($2^l$moons$^U$). Instead, in both conditions they prefer a simple single vertical or horizontal decision boundary, as Table 2 shows[2].

**2.** With two labeled points, even if they are explicitly given the graph structure in the form of highlighting, participants still do not perform manifold learning (comparing $2^l$moons$^U$ vs. $2^l$moons$^U$h). It seems they are "blocked" by the simpler vertical or horizontal hypothesis, which perfectly explains the labeled data.

**3.** When there are four labeled points but no highlighting, the distribution of unlabeled data still does not encourage people to perform manifold learning (comparing $4^l$grid$^U$ vs. $4^l$moons$^U$). This further suggests that people can not easily extract manifold structure from unlabeled data in order to learn, when there is no hint to do so. However, most participants have given up the simple single vertical or horizontal decision boundary, because it contradicts with the four labeled points.

**4.** Finally, when we provide the graph structure, there is a marked switch to manifold learning (comparing $4^l$moons$^U$ vs. $4^l$moons$^U$h). This suggests that a combination of the elimination of preferred, simpler hypotheses, together with a stronger graph hint, finally gives the originally less preferred manifold learning model a chance of being used. It is under this condition that we observed human manifold learning behavior.

# 5 Humans do not Blindly Follow the Highlighting

Do humans really learn using manifolds? Could they have adopted a "follow-the-highlighting" procedure to label the manifolds 100% correctly: in the beginning, click on a labeled card $x$ to highlight its neighboring unlabeled cards; pick one such neighbor $x'$ and classify it with the label of $x$; now click on (the now labeled) $x'$ to find one of its unlabeled neighbors $x''$, and repeat? Because our graph has disconnected components with consistently labeled seeds, this procedure will succeed. The procedure is known as propagating-1NN in semi-supervised learning (Algorithm 2.7, [11]). In this section we present three arguments that humans are not blindly following the highlighting.

First, participants in $2^l\text{moons}^\text{U}\text{h}$ did not learn the manifold while those in $4^l\text{moons}^\text{U}\text{h}$ did, even though the two conditions have the same $\epsilon$NN highlighting.

Second, a necessary condition for follow-the-highlighting is to always classify an unlabeled $x'$ according to a labeled highlighted neighbor $x$. Conversely, if a participant classifies $x'$ as class $y'$, while all neighbors of $x'$ are either still unlabeled or have labels other than $y'$, she could not have been using follow-the-highlighting on $x'$. We say she has taken a leap-of-faith on $x'$. The $4^l\text{moons}^\text{U}\text{h}$ participants had an average of 17 leaps-of-faith among about 78 classifications[3], while strict follow-the-highlighting procedure would yield zero leaps-of-faith.

Third, the basic challenge of follow-the-highlighting is that the underlying manifold structure of the stimuli may have been irrelevant. Would participants have shown the same behavior, following the highlighting, regardless of the actual stimuli? We therefore designed the following experiment. Take the $4^l\text{moons}^\text{U}\text{h}$ graph which has 4 labeled nodes, 78 unlabeled nodes, and an adjacency matrix (i.e., edges) defined by $\epsilon$NN, as shown in Figure 3. Take a random permutation $\pi = (\pi_1, \ldots, \pi_{78})$. Map the feature vector of the $i$th unlabeled point to $x_{\pi_i}$, while keeping the adjacency matrix the same. This creates the random-looking graph in Figure 6(a) which we call $4^l\text{moons}^\text{U}\text{h}^\text{R}$ condition (the suffix R stands for random), which is equivalent to the $4^l\text{moons}^\text{U}\text{h}$ graph in structure. In particular, there are two connected components with consistent labeled seeds. However, now the highlighted neighbors may look very different than the clicked card.

If we assume humans blindly follow the highlighting (perhaps noisily), then we predict that they are more likely to classify those unlabeled points nearer (in shortest path length on the graph, not Euclidean distance) a labeled point with the latter's label; and that this correlation should be the same under $4^l\text{moons}^\text{U}\text{h}^\text{R}$ and $4^l\text{moons}^\text{U}\text{h}$. This prediction turns out to be false. 30 additional undergraduates participated in the new $4^l\text{moons}^\text{U}\text{h}^\text{R}$ condition. Figure 6(b) shows the above behavioral evaluation, which does not exhibit the predicted correlation, and is clearly different from the same evaluation for $4^l\text{moons}^\text{U}\text{h}$ in Figure 6(c). Again, this is evidence that humans are not just following the highlighting. In fact, human behavior in $4^l\text{moons}^\text{U}\text{h}^\text{R}$ is similar to $4^l\text{moons}^\text{U}$. That is, having random highlighting is similar to having no highlighting in how it affects human categorization. This can be seen from the last rows of Tables 1 and 2, and Figure 6(d)[4].

# 6 Discussion

We have presented a set of experiments exploring human manifold learning behaviors. Our results suggest that people can perform manifold learning, but only when there is no alternative, simpler explanation of the data, and people need strong hints about the graph structure.

We propose that Bayesian model selection is one possible way to explain these human behaviors. Recall we defined seven Gaussian Processes, each with a different kernel. For a given GP with kernel $k$, the evidence $p(y_{1:l} \mid x_{1:l}, k)$ is the marginal likelihood on labeled data, integrating out the hidden discriminant function sampled from the GP. With multiple candidate GP models, one may perform model selection by selecting the one with the largest marginal likelihood. From the absence of manifold learning in conditions without highlighting or with random highlighting, we speculate that the GP with the graph-based kernel $\tilde{k}$ (1) is special: it is accessible in a participant's repertoire

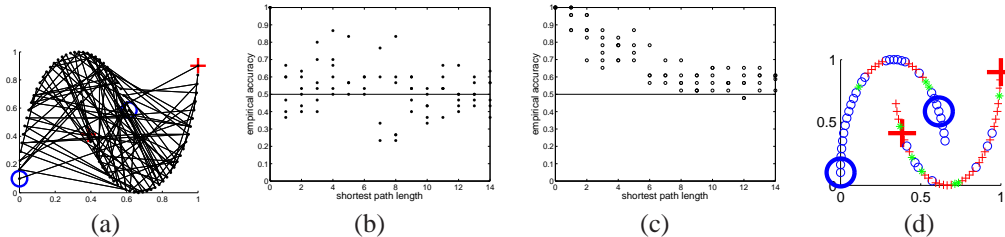

Figure 6: The $4^l\mathrm{moons}^\mathrm{U}\mathrm{h}^\mathrm{R}$ experiment with 30 participants. (a) The $4^l\mathrm{moons}^\mathrm{U}\mathrm{h}^\mathrm{R}$ condition. (b) The behavioral evaluation for $4^l\mathrm{moons}^\mathrm{U}\mathrm{h}^\mathrm{R}$, where the $x$-axis is the shortest path length of an unlabeled point to a labeled point, and the $y$-axis is the fraction of participants who classified that unlabeled point consistent with the nearest labeled point. (c) The same behavioral evaluation for $4^l\mathrm{moons}^\mathrm{U}\mathrm{h}$. (d) The majority vote in $4^l\mathrm{moons}^\mathrm{U}\mathrm{h}^\mathrm{R}$.

only when strong hints (highlighting) exists and agrees with the underlying unlabeled data manifold structure. Under this assumption, we can then explain the contrast between the lack of manifold learning in $2^l\mathrm{moons}^\mathrm{U}\mathrm{h}$, and the presence of manifold learning in $4^l\mathrm{moons}^\mathrm{U}\mathrm{h}$. On one hand, for the $2^l\mathrm{moons}^\mathrm{U}\mathrm{h}$ condition, the evidence for the seven GP models on the two labeled points are: (graph) 0.249, (1NN,$\ell_2$) 0.250, (1NN,$\ell_1$) 0.250, (multi-v) 0.250, (multi-h) 0.250, (single-v) 0.249, (single-h) 0.249. The graph-based GP has slightly lower evidence than several other GPs, which may be due to our specific choice of kernel parameters in (1). In any case, there is no reason to prefer the GP with a graph kernel, and we do not expect humans to learn on manifold in $2^l\mathrm{moons}^\mathrm{U}\mathrm{h}$. On the other hand, for $4^l\mathrm{moons}^\mathrm{U}\mathrm{h}$, the evidence for the seven GP models on those four labeled points are: (graph) 0.0626, (1NN,$\ell_2$) 0.0591, (1NN,$\ell_1$) 0.0625, (multi-v) 0.0625, (multi-h) 0.0625, (single-v) 0.0341, (single-h) 0.0342. The graph-based GP has a small lead over other GPs. In particular, it is better than the evidence 1/16 for kernels that treat the four labeled points essentially independently. The graph-based GP obtains this lead by warping the space along the two manifolds so that the two positive (resp. negative) labeled points tend to co-vary. Thus, there is a reason to prefer the GP with a graph kernel, and we do expect humans to learn on manifold in $4^l\mathrm{moons}^\mathrm{U}\mathrm{h}$.

We also explore the convex combination of the seven GPs as a richer model for human behavior: $k(\lambda) = \sum_{i=1}^{7} \lambda_i k_i$, where $\lambda_i \geq 0, \sum_i \lambda_i = 1$. This allows a weighted combination of kernels to be used, and is more powerful than selecting a single kernel. Again, we optimize the mixing weights $\lambda$ by maximizing the evidence $p(y_{1:l} \mid x_{1:l}, k(\lambda))$. This is a constrained optimization problem, and can be easily solved up to local optimum (because evidence is in general non-convex) with a projected gradient method, given the gradient of the log evidence. For the $2^l\mathrm{moons}^\mathrm{U}\mathrm{h}$ condition, in 100 trials with random starting $\lambda$ values, the maximum evidence always converges to 1/4, while the optimum $\lambda$ is not unique and occupies a subspace $(0, \lambda_2, \lambda_3, \lambda_4, \lambda_5, 0, 0)$ with $\lambda_2 + \lambda_3 + \lambda_4 + \lambda_5 = 1$ and mean $(0, 0.27, 0.25, 0.22, 0.26, 0, 0)$. Note the weight for the graph-based kernel $\lambda_1$ is zero. In contrast, for the $4^l\mathrm{moons}^\mathrm{U}\mathrm{h}$ condition, in 100 trials $\lambda$ overwhelmingly converges to $(1, 0, 0, 0, 0, 0, 0)$ with evidence 0.0626. i.e., it again suggests that people would perform manifold learning in $4^l\mathrm{moons}^\mathrm{U}\mathrm{h}$.

Of course, this Bayesian model selection analysis is over-simplified. For instance, we did not consider people's prior $p(\lambda)$ on GP models, i.e., which model they would prefer before seeing the data. It is possible that humans favor models which produce axis-parallel decision boundaries. Defining and incorporating non-uniform $p(\lambda)$ priors is a topic for future research.

**Acknowledgments** We thank Rob Nowak and the anonymous reviewers for their valuable comments that motivated us to conduct the new experiments discussed in Section 5 after initial review. This work is supported in part by NSF IIS-0916038, NSF IIS-0953219, NSF DRM/DLS-0745423, and AFOSR FA9550-09-1-0313.

## Footnotes

[1]The condition $4^l$moons$^U$h$^R$ will be explained later in Section 5.

[2]The two rows in Table 1 for these two conditions are therefore misleading, as it averages classification made with vertical and horizontal decision boundaries. Also note that in the $2^l$ conditions (multi-v) and (multi-h) are effectively single linear boundary models (see Figure 4) and differ from (single-v) and (single-h) only slightly due to the training method used.

[3]The individual number of leaps-of-faith are 0, 1, 2, 4, 10, 13, 13, 14, 14, 15, 15, 16, 18, 19, 20, 21, 22, 24, 25, 27, 33, 36, and 36 respectively, for the 23 participants.

[4]In addition, if we create a GP from the Laplacian of the random highlighting graph, the GP accuracy in predicting $4^l\text{moons}^\text{U}\text{h}^\text{R}$ human majority vote is 0.46, and the percentage of participants in $4^l\text{moons}^\text{U}\text{h}^\text{R}$ who can be attributed to this model is 0.

## References

[1] Mikhail Belkin, Partha Niyogi, and Vikas Sindhwani. Manifold regularization: A geometric framework for learning from labeled and unlabeled examples. *Journal of Machine Learning Research*, 7:2399–2434, November 2006.

[2] Olivier Chapelle, Bernhard Schölkopf, and Alexander Zien, editors. *Semi-supervised learning*. MIT Press, 2006.

[3] R. M. Nosofsky. Attention, similarity, and the identification-categorization relationship. *Journal of Experimental Psychology: General*, 115(1):39–57, 1986.

[4] Carl E. Rasmussen and Christopher K. I. Williams. *Gaussian Processes for Machine Learning*. MIT Press, 2006.

[5] Carl E. Rasmussen and Christopher K. I. Williams. GPML matlab code, 2007. http://www.gaussianprocess.org/gpml/code/matlab/doc/, accessed May, 2010.

[6] Vikas Sindhwani, Partha Niyogi, and Mikhail Belkin. Beyond the point cloud: from transductive to semi-supervised learning. In *ICML05, 22nd International Conference on Machine Learning*, 2005.

[7] Katleen Vandist, Maarten De Schryver, and Yves Rosseel. Semisupervised category learning: The impact of feedback in learning the information-integration task. *Attention, Perception, & Psychophysics*, 71(2):328–341, 2009.

[8] Guy Wallis and Heinrich H. Bülthoff. Effects of temporal association on recognition memory. *Proceedings of the National Academy of Sciences*, 98(8):4800–4804, 2001.

[9] Dengyong Zhou, Olivier Bousquet, Thomas Lal, Jason Weston, and Bernhard Schḷkopf. Learning with local and global consistency. In *Advances in Neural Information Processing System 16*, 2004.

[10] Xiaojin Zhu, Zoubin Ghahramani, and John Lafferty. Semi-supervised learning using Gaussian fields and harmonic functions. In *The 20th International Conference on Machine Learning (ICML)*, 2003.

[11] Xiaojin Zhu and Andrew B. Goldberg. *Introduction to Semi-Supervised Learning*. Synthesis Lectures on Artificial Intelligence and Machine Learning. Morgan & Claypool Publishers, San Rafael, CA, 2009.

